# Connection Topology and Dynamics in Lateral Inhibition Networks

C. M. Marcus, F. R. Waugh, and R. M. Westervelt
Department of Physics and Division of Applied Sciences, Harvard University
Cambridge, MA 02138

## ABSTRACT

We show analytically how the stability of two-dimensional lateral inhibition neural networks depends on the local connection topology. For various network topologies, we calculate the critical time delay for the onset of oscillation in continuous-time networks and present analytic phase diagrams characterizing the dynamics of discrete-time networks.

## 1  INTRODUCTION

Mutual inhibition in an array of neurons is a common feature of sensory systems including vision, olfaction, and audition in organisms ranging from invertebrates to man. A well-studied instance of this configuration is lateral inhibition between neighboring photosensitive neurons in the retina (Dowling, 1987). Inhibition serves in this case to enhance the perception of edges and to broaden the dynamic range by setting a local reference point for measuring intensity variations. Lateral inhibition thus constitutes the first stage of visual information processing. Many artificial vision systems also take advantage of the computational power of lateral inhibition by directly wiring inhibition into the photodetecting electronic hardware (Mead, 1989).

Lateral inhibition may create extensive feedback paths, leading to network-wide collective oscillations. Sustained oscillations arising from lateral inhibition have been observed in biological visual systems—specifically, in the compound eye of the horseshoe crab *Limulus* (Barlow and Fraioli, 1978; Coleman and Renninger, 1978)—as well as in artificial vision systems, for instance plaguing an early version of the electronic retina chip built by Mead *et al.* (Wyatt and Standley, 1988; Mead, 1989).

In this paper we study the dynamics of simple neural network models of lateral inhibition in a variety of two-dimensional connection schemes. The lattice structures we study are shown in Fig. 1. Two-dimensional lattices are of particular importance to artificial vision systems because they allow an efficient mapping of an image onto a network and because they are well-suited for implementation in VLSI circuitry. We show that the

stability of these networks depends sensitively on such design considerations as local connection topology, neuron self-coupling, the steepness or gain of the neuron transfer function, and details of the network dynamics such as connection delays for continuous-time dynamics or update rule for discrete-time dynamics.

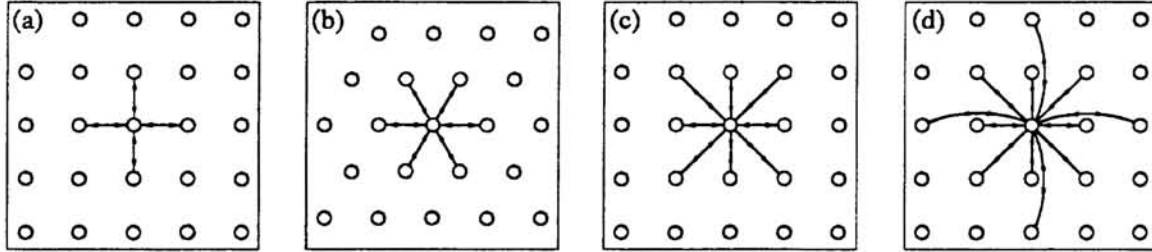

**Figure 1:** Connection schemes for two-dimensional lateral inhibition networks considered in this paper: (a) nearest-neighbor connections on a square lattice; (b) nearest-neighbor connections on a triangular lattice; (c) 8-neighbor connections on a square lattice; and (d) 12-neighbor connections on a square lattice.

The paper is organized as follows. Section 2 introduces the dynamical equations describing continuous-time and discrete-time lateral inhibition networks. Section 3 discusses the relationship between lattice topology and critical time delay for the onset of oscillation in the continuous-time case. Section 4 presents analytic phase diagrams characterizing the dynamics of discrete-time lateral inhibition networks as neuron gain, neuron self-coupling, and lattice structure are varied. Our conclusions are presented in Section 5.

## 2    NETWORK DYNAMICS

We begin by considering a general neural network model defined by the set of electronic circuit equations

$$C_i \, du_i(t')/dt' = -u_i(t')/R_i + \sum_j T_{ij} f_j\left(u_j\left(t' - \tau_{ij}'\right)\right) + I_i \quad , \quad i = 1, \dots, N, \qquad (1)$$

where $u_i$ is the voltage, $C_i$ the capacitance, and $R_i^{-1} = \sum_j |T_{ij}|$ the total conductance at the input of neuron $i$. Input to the network is through the applied currents $I_i$. The nonlinear transfer function $f_i$ is taken to be sigmoidal with odd symmetry and maximum slope at the origin. A time delay $\tau_{ij}'$ in the communication from neuron $i$ to neuron $j$ has been explicitly included. Such a delay could arise from the finite operating speed of the elements—neurons or amplifiers—or from the finite propagation speed of the interconnections. For the case of lateral inhibition networks with self-coupling, the connection matrix is given by

$$T_{ij} = \begin{cases} \gamma & \text{for } i = j \\ -1 & \text{for } i, j \text{ connected neighbors} \\ 0 & \text{otherwise,} \end{cases} \qquad (2)$$

which makes $R_i^{-1} = |\gamma| + z$ for all $i$, where $z$ is the number of connected neighbors. For simplicity, we take all neurons to have the same delay and characteristic relaxation

time ($\tau_i' = \tau_{delay}$, $R_i C_i = \tau_{relax}$ for all $i$) and identical transfer functions. With these assumptions, Eq. (1) can be rescaled and written in terms of the neuron outputs $x_i(t)$ as

$$dx_i(t)/dt = -x_i(t) + F\left(\sum_j T_{ij} x_j(t - \tau) + I_i\right), \quad i = 1, ..., N, \tag{3}$$

where the odd, sigmoidal function $F$ now appears outside the sum. The function $F$ is characterized by a maximum slope $\beta$ ($> 0$), and its saturation amplitude can be set to $\pm 1$ without loss of generality. The commonly used form $F(h) = \tanh(\beta h)$ satisfies these requirements; we will continue to use $F$ to emphasize generality. As a result of rescaling, the delay time $\tau$ is now measured in units of network relaxation time (i.e. $\tau = \tau_{delay}/\tau_{relax}$), and the connection matrix is normalized such that $\sum_j |T_{ij}| = 1$ for all $i$. Stability of Eq. (3) against coherent oscillation will be discussed in Section 3.

The discrete-time iterated map,

$$x_i(t+1) = F\left(\sum_j T_{ij} x_j(t) + I_i\right), \quad i = 1, ..., N, \tag{4}$$

with parallel updating of neuron states $x_i(t)$, corresponds to the long-delay limit of Eq. (3) (care must be taken in considering this limit; not all aspects of the delay system carry over to the map (Mallet-Paret and Nussbaum, 1986)). The iterated map network, Eq. (4), is particularly useful for implementing fast, parallel networks using conventional computer clocking techniques. The speed advantage of parallel dynamics, however, comes at a price: the parallel-update network may oscillate even when the corresponding sequential update network is stable. Section 4 gives phase diagrams based on global stability analysis which explicitly define the oscillation-free operating region of Eq. (4) and its generalization to a multistep updating rule.

## 3    STABILITY OF LATTICES WITH DELAYED INHIBITION

In the absence of delay ($\tau = 0$) the continuous-time lateral inhibition network, Eq. (3), always converges to a fixed point attractor. This follows from the famous stability criterion based on a Liapunov (or "energy") function (Cohen and Grossberg, 1983; Hopfield, 1984), and relies on the symmetry of the lateral inhibitory connections (i.e. $T_{ij} = T_{ji}$ for all connection schemes in Fig. 1). This guarantee of convergence does not hold for nonzero delay, however, and it is known that adding delay can induce sustained, coherent oscillation in a variety of symmetrically connected network configurations (Marcus and Westervelt, 1989a). Previously we have shown that certain delay networks of the form of Eq. (3)—including lateral inhibition networks—will oscillate coherently, that is with all neurons oscillating in phase, for sufficiently large delay. As the delay is reduced, however, the oscillatory mode becomes unstable, leaving only fixed point attractors. A critical value of delay $\tau_{crit}$ below which sustained oscillation vanishes for any value of neuron gain $\beta$ is given by

$$\tau_{crit} = -ln(1 + \lambda_{max}/\lambda_{min}) \qquad (0 < \lambda_{max} < -\lambda_{min}) \tag{5}$$

where $\lambda_{max}$ and $\lambda_{min}$ are the extremal eigenvalues of the connection matrix $T_{ij}$. The analysis leading to (5) is based on a local stability analysis of the coherent oscillatory mode. Though this local analysis lacks the rigor of a global analysis (which can be done for $\tau = 0$ and for the discrete-time case, Eq. (4)) the result agrees well with experiments and numerical simulations (Marcus and Westervelt, 1989a).

It is straightforward to find the spectrum of eigenvalues for the lattices in Fig. 1. Assuming periodic boundary conditions, one can expand the eigenvalue equation $Tx = \lambda x$ in terms of periodic functions $x_j = x_o \, exp(i\, q \cdot R_j)$, where $R_j$ is the 2D vector position of neuron $j$ and $q$ is the reciprocal lattice vector characterizing a particular eigenmode. In the large network limit, this expansion leads to the following results for the square and triangular lattices with nearest neighbor connections and self-connection $\gamma$ [see next section for a table of eigenvalues]:

$$\tau_{crit} \rightarrow ln(1/2 - 2/\gamma) \qquad (-4 < \gamma < 0) \qquad \text{[n.n. square lattice, Fig. 1(a)] ,} \qquad (6a)$$

$$\tau_{crit} \rightarrow ln[(\gamma - 6)/(2\gamma - 3)] \quad (-3 < \gamma < 3/2) \quad \text{[n.n. triangular lattice, Fig. 1(b)].} \quad (6b)$$

Curves showing $\tau_{crit}$ as a function of self-connection $\gamma$ are given in Fig. 2. These reveal the surprising result that *the triangular lattice is much more prone to delay-induced oscillation than the square lattice*. For instance, with no self connection ($\gamma = 0$), the square lattice does not show sustained oscillation for any finite delay, while the triangular lattice oscillates for $\tau > \ln 2 \cong 0.693$.

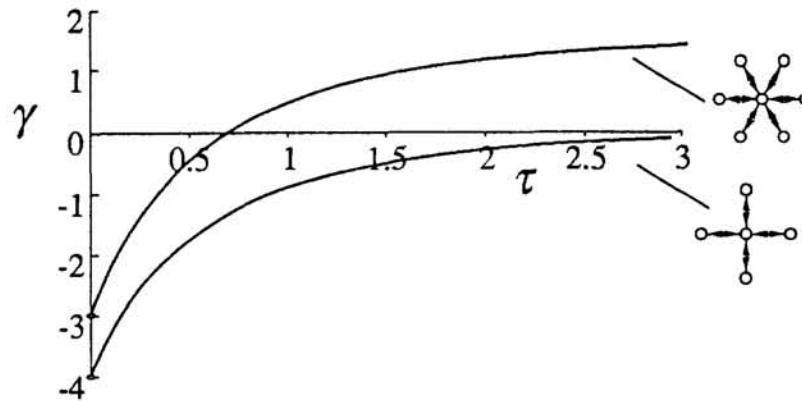

**Figure 2:** Critical delay $\tau_{crit}$ as a function of self-connection $\gamma$, from Eq. (6). Note that for $\gamma = 0$ only triangular lattice oscillates at finite delay. The analysis does not apply at exactly $\tau = 0$, where both networks are stable for all values of $\gamma$.

The important difference between these two lattices—and the quality which accounts for their dissimilar stability properties—is *not* simply the number of neighbors, but is the presence of *frustration* in the triangular lattice but not in the square lattice. Lateral inhibition, like antiferromagnetism, forms closed loops in the triangular lattice which do not allow all of the connections to be satisfied by any arrangement of neuron states. In contrast, lateral inhibition on the square lattice is not frustrated, and is, in fact, exactly equivalent to lateral excitation via a gauge transformation. We note that a similar situation exists in 2D magnetic models: while models of 2D ferromagnetism on square and triangular lattices behave nearly identically (both are nonfrustrated), the corresponding 2D antiferromagnets are quite different, due to the presence of frustration in the triangular lattice, but not the square lattice (Wannier, 1950).

## 4    LATTICES WITH ITERATED-MAP DYNAMICS

Next we consider lateral inhibition networks with discrete-time dynamics where all neuron states are updated in parallel. The standard parallel dynamics formulation was given above as Eq. (4), but here we will consider a generalized updating rule which offers some important practical advantages. The generalized system we consider updates the neuron states based on an average over $M$ previous time steps, rather than just using a single previous state to generate the next. This multistep rule is somewhat like including time delay, but as we will see, increasing $M$ actually makes the system more stable compared to standard parallel updating. This update rule also differs from the delay-differential system in permitting a rigorous global stability analysis. The dynamical system we consider is defined by the following set of coupled iterated maps:

$$x_i(t+1) = F\left(\sum_j T_{ij}z_j(t) + I_i\right) ; \qquad z_j(t) = M^{-1}\sum_{\tau=0}^{M-1} x_j(t-\tau) , \qquad (7)$$

where $i,j = 1,...,N$ and $M \in \{1,2,3,...\}$. The standard parallel updating rule, Eq.(4), is recovered by setting $M = 1$.

A global analysis of the dynamics of Eq. (7) for any symmetric $T_{ij}$ is given in (Marcus and Westervelt, 1990), and for $M=1$ in (Marcus and Westervelt, 1989b). It is found that for any $M$, if all eigenvalues $\lambda$ satisfy $\beta|\lambda|<1$ then there is a single attractor which depends only on the inputs $I_i$. For $I_i = 0$, this attractor is the origin, i.e. all neurons at zero output. Whenever $\beta|\lambda|>1$ for one or more eigenvalues, multiple fixed points as well as periodic attractors may exist. There is, in addition, a remarkably simple global stability criterion associated with Eq. (7): satisfying the condition $1/\beta > -\lambda_{min}\left(T_{ij}\right)/M$ insures that no periodic attractors exist, though there may be a multiplicity of fixed point attractors. As in the previous section, $\lambda_{min}$ is the most negative eigenvalue of $T_{ij}$. If $T_{ij}$ has no negative eigenvalues, then $\lambda_{min}$ is the smallest positive eigenvalue, and the stability criterion is satisfied trivially since $\beta$ is defined to be positive.

These stability results may be used to compute analytic phase diagrams for the various connection schemes shown in Fig. 1 and defined in Eq. (3). The extremal eigenvalues of $T_{ij}$ are calculated using the Fourier expansion described above. In the limit of large lattice size and assuming periodic boundary conditions, we find the following:

|  | square n. n. | triangle n. n. | square 8 - n. | square 12 - n. |
|---|---|---|---|---|
| $\lambda_{max}$: | $\dfrac{\gamma+4}{\|\gamma\|+4}$ | $\dfrac{\gamma+3}{\|\gamma\|+6}$ | $\dfrac{\gamma+4}{\|\gamma\|+8}$ | $\dfrac{\gamma+13/3}{\|\gamma\|+12}$ |
| $\lambda_{min}$: | $\dfrac{\gamma-4}{\|\gamma\|+4}$ | $\dfrac{\gamma-6}{\|\gamma\|+6}$ | $\dfrac{\gamma-8}{\|\gamma\|+8}$ | $\dfrac{\gamma-12}{\|\gamma\|+12}$ |

The resulting phase diagrams characterizing regions with different dynamic properties are shown in Fig. 3. The four regions indicated in the diagrams are characterized as follows: (1) *orig*: low gain regime where a unique fixed point attractor exists (that attractor is the origin for $I_i = 0$ ); (2) *fp*: for some inputs $I_i$ multiple fixed point attractors may exist, each with an attracting basin, *but no oscillatory attractors exist in this region (i.e. no attractors with period >1)*; (3) *osc*: at most one fixed point attractor, but one or more oscillatory modes also may exist; (4) *fp + osc*: multiple fixed points as well as oscillatory attractors may exist.

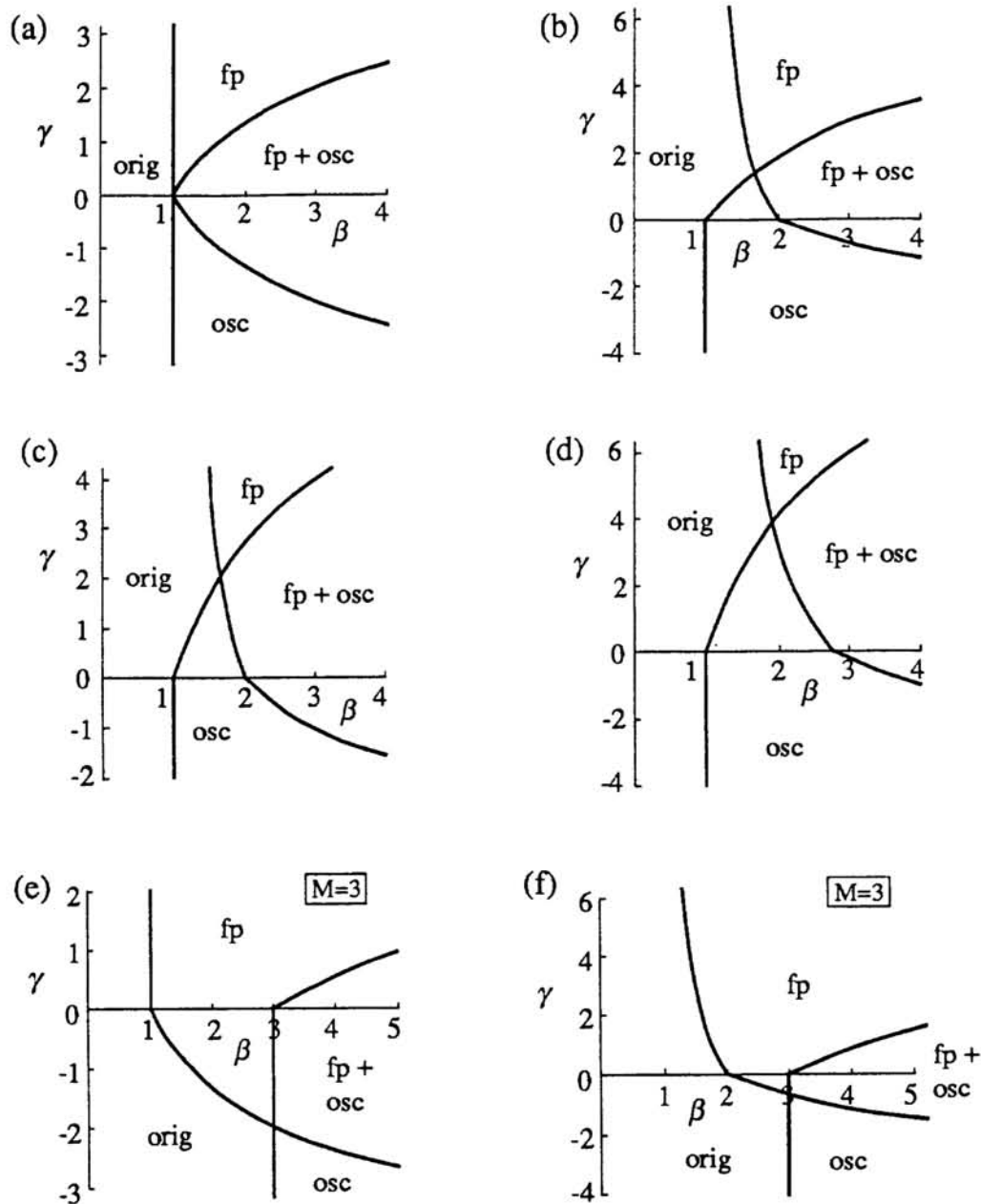

**Figure 3**: Phase diagrams based on global analysis for lateral inhibition networks with discrete-time parallel dynamics [Eq.(7)] as a function of neuron gain $\beta$ and self-connection $\gamma$. Regions *orig*, *fp*, *osc*, and *fp+osc* are defined in text. (a) Nearest-neighbor connections on a square lattice and single-step updating ($M=1$); (b) nearest-neighbor connections on a triangular lattice, $M=1$; (c) 8-neighbor connections on a square lattice, $M=1$; (d) 12-neighbor connections on a square lattice, $M=1$; (e) nearest-neighbor connections on a square lattice, $M=3$; (f) nearest-neighbor connections on a triangular lattice, $M=3$.

## 5    CONCLUSIONS

We have shown analytically how the dynamics of two-dimensional neural network models of lateral inhibition depends on both single-neuron properties—such as the slope of the sigmoidal transfer function, delayed response, and the strength of self-connection—and also on the topological properties of the network.

The design rules implied by the analysis are in some instances what would be expected intuitively. For example, the phase diagrams in Fig. 4 show that in order to eliminate oscillations one can either include a positive self-connection term or decrease the gain of the neuron. It is also not surprising that reducing the time delay in a delay-differential system eliminates oscillation. Less intuitive is the observation that for discrete-time dynamics using a multistep update rule greatly expands the region of oscillation-free operation (compare, for example Figs. 4(a) and 4(e)). One result emerging in this paper that seems quite counterintuitive is the dramatic effect of connection topology, which persists even in the limit of large lattice size. This point was illustrated in a comparison of networks with delayed inhibition on square and triangular lattices, where it was found that in the absence of self-connection, only the triangular lattices will show sustained oscillation.

Finally, we note that it is not clear to us how to generalize our results to other network models, for example to models with *asymmetric* connections which allow for direction-selective motion detection. Such questions remain interesting challenges for future work.

### Acknowledgments

We thank Bob Meade and Cornelia Kappler for informative discussions. One of us (C.M.M.) acknowledges support as an IBM Postdoctoral Fellow, and one (F.R.W.) from the Army Research Office as a JSEP Graduate Fellow. This work was supported in part by ONR contract N00014-89-J-1592, JSEP contract N00014-89-J-1023, and DARPA contract AFOSR-89-0506.

### References

Barlow, R. B. and A. J. Fraioli (1978), J. Gen. Physiol., **71**, 699.
Cohen, M. A., and S. Grossberg (1983), IEEE Trans. SMC-13, 815.
Coleman, B. D. and G.H. Renninger (1978), Math. Biosc. **38**, 123.
Dowling, J. E. (1987), *The Retina: An Approachable Part of the Brain* (Harvard University Press, Cambridge, MA).
Hopfield, J. J. (1984), Proc. Nat. Acad. Sci. USA **81**, 3008.
Mallet-Paret, J. and R. D. Nussbaum (1986) in *Chaotic Dynamics and Fractals*, edited by M. F. Barnsley and S. G. Demko, (Academic Press, Orlando) p. 263.
Marcus, C. M. and R. M. Westervelt (1989a), Phys. Rev. A **39**, 347.
Marcus, C. M. and R. M. Westervelt (1989b), Phys. Rev. A **40**, 501.
Marcus, C. M. and R. M. Westervelt (1990), Phys. Rev. A **42**, 2410.
Mead, Carver A. (1989), *Analog VLSI and Neural Systems* (Addison-Wesley, Reading, MA).
Wyatt, Jr., J. L., and D. L. Standley (1988), in *Neural Information Processing Systems, Denver CO, 1987*, edited by D. Z. Anderson, (AIP, New York), p. 860.
Wannier, G. M. (1950), Phys. Rev. **79**, 357.